# Unsupervised Feature Selection for the $k$-means Clustering Problem

**Christos Boutsidis**
Department of Computer Science
Rensselaer Polytechnic Institute
Troy, NY 12180
boutsc@cs.rpi.edu

**Michael W. Mahoney**
Department of Mathematics
Stanford University
Stanford, CA 94305
mmahoney@cs.stanford.edu

**Petros Drineas**
Department of Computer Science
Rensselaer Polytechnic Institute
Troy, NY 12180
drinep@cs.rpi.edu

## Abstract

We present a novel feature selection algorithm for the $k$-means clustering problem. Our algorithm is randomized and, assuming an accuracy parameter $\epsilon \in (0, 1)$, selects and appropriately rescales in an unsupervised manner $\Theta(k \log(k/\epsilon)/\epsilon^2)$ features from a dataset of arbitrary dimensions. We prove that, if we run any $\gamma$-approximate $k$-means algorithm ($\gamma \geq 1$) on the features selected using our method, we can find a $(1 + (1 + \epsilon)\gamma)$-approximate partition with high probability.

## 1 Introduction

Clustering is ubiquitous in science and engineering, with numerous and diverse application domains, ranging from bioinformatics and medicine to the social sciences and the web [15]. Perhaps the most well-known clustering algorithm is the so-called "$k$-means" algorithm or Lloyd's method [22], an iterative expectation-maximization type approach, which attempts to address the following objective: given a set of points in a Euclidean space and a positive integer $k$ (the number of clusters), split the points into $k$ clusters so that the total sum of the (squared Euclidean) distances of each point to its nearest cluster center is minimized. This optimization objective is often called the $k$-means clustering objective. (See Definition 1 for a formal discussion of the $k$-means objective.) The simplicity of the objective, as well as the good behavior of the associated algorithm (Lloyd's method [22, 28]), have made $k$-means enormously popular in applications [32].

In recent years, the high dimensionality of the modern massive datasets has provided a considerable challenge to $k$-means clustering approaches. First, the curse of dimensionality can make algorithms for $k$-means clustering very slow, and, second, the existence of many irrelevant features may not allow the identification of the relevant underlying structure in the data [14]. Practitioners addressed such obstacles by introducing *feature selection* and *feature extraction* techniques. It is worth noting that feature selection selects a small subset of actual features from the data and then runs the clustering algorithm only on the selected features, whereas feature extraction constructs a small set of artificial features and then runs the clustering algorithm on the constructed features. Despite the significance of the problem, as well as the wealth of heuristic methods addressing it (see Section 3), there exist no provably accurate feature selection methods and extremely few provably accurate feature extraction methods for the $k$-means clustering objective (see Section 3.1 for the later case).

Our work here addresses this shortcoming by presenting the first provably accurate feature selection algorithm for $k$-means clustering. Our algorithm constructs a probability distribution for the feature space, and then selects a small number of features (roughly $k \log(k)$, where $k$ is the number of clusters) with respect to the computed probabilities. (See Section 2 for a detailed description of our algorithm.) Then, we argue that running $k$-means clustering algorithms on the selected features returns a constant-factor approximate partition to the optimal. (See Theorem 1 in Section 2.)

We now formally define the $k$-means clustering problem using the so-called cluster indicator matrix. Also, recall that the Frobenius norm of a matrix (denoted by $\|\cdot\|_F$) is equal to the square root of the sum of the squares of its elements. (See also Section 4.1 for useful notation.)

**Definition 1** [THE K-MEANS CLUSTERING PROBLEM]
*Given a matrix $A \in \mathbb{R}^{n \times d}$ (representing $n$ points – rows – described with respect to $d$ features – columns) and a positive integer $k$ denoting the number of clusters, find the $n \times k$ indicator matrix $X_{opt}$ such that*

$$X_{opt} = \arg \min_{X \in \mathcal{X}} \left\| A - XX^T A \right\|_F^2. \tag{1}$$

*The optimal value of the k-means clustering objective is*

$$F_{opt} = \min_{X \in \mathcal{X}} \left\| A - XX^T A \right\|_F^2 = \left\| A - X_{opt} X_{opt}^T A \right\|_F^2. \tag{2}$$

*In the above $\mathcal{X}$ denotes the set of all $n \times k$ indicator matrices $X$.*

We briefly expand on the notion of an $n \times k$ indicator matrix $X$. Such matrices have exactly one non-zero element per row, which denotes cluster membership. Equivalently, for all $i = 1, \ldots, n$ and $j = 1, \ldots, k$, the $i$-th row (point) of $A$ belongs to the $j$-th cluster if and only if $X_{ij}$ is non-zero; in particular $X_{ij} = 1/\sqrt{s_j}$, where $s_j$ is the number of points in the corresponding cluster (i.e. the number of non-zero elements in the $j$-th column of $X$). Note that the columns of $X$ are normalized and pairwise orthogonal so that their Euclidean norm is equal to one, and $X^T X = I_k$, where $I_k$ is the $k \times k$ identity matrix. An example of such an indicator matrix $X$ representing three points (rows in $X$) belonging to two different clusters (columns in $X$) is given below; note that the points corresponding to the first two rows of $X$ belong to the first cluster ($s_1 = 2$) and the other point to the second cluster ($s_2 = 1$):

$$X = \begin{pmatrix} 1/\sqrt{2} & 0 \\ 1/\sqrt{2} & 0 \\ 0 & 1/\sqrt{1} \end{pmatrix}.$$

The above definition of the $k$-means objective is exactly equivalent with the standard definition of $k$-means clustering [28]. To see this notice that $\left\| A - XX^T A \right\|_F^2 = \sum_{i=1}^n \|A_{(i)} - X_{(i)} X^T A\|_2^2$, while for $i = 1, ..., n$, $X_{(i)} X^T A$ denotes the centroid of the cluster the $i$-th point belongs to. In the above, $A_{(i)}$ and $X_{(i)}$ denote the $i$-th rows of $A$ and $X$, respectively.

## 2   The feature selection algorithm and the quality-of-clustering results

Algorithm 1 takes as inputs the matrix $A \in R^{n \times d}$, the number of clusters $k$, and an accuracy parameter $\epsilon \in (0,1)$. It first computes the top-$k$ right singular vectors of $A$ (columns of $V_k \in R^{d \times k}$). Using these vectors, it computes the so-called (normalized) leverage scores [4, 24]; for $i = 1, ..., d$ the $i$-th leverage score equals the square of the Euclidian norm of the $i$-th row of $V_k$ (denoted by $(V_k)_{(i)}$). The $i$-th leverage score characterizes the importance of the $i$-th feature with respect to the $k$-means objective. Notice that these scores (see the definition of $p'_i s$ in step 2 of Algorithm 1) form a probability distribution over the columns of $A$ since $\sum_{i=1}^n p_i = 1$. Then, the algorithm chooses a sampling parameter $r$ that is equal to the number of (rescaled) features that we want to select. In order to prove our theoretical bounds, $r$ should be fixed to $r = \Theta(k \log(k/\epsilon)/\epsilon^2)$ at this step (see section 4.4). In practice though, a small value of $r$, for example $r = 10k$, seems sufficient (see section 5). Having $r$ fixed, Algorithm 1 performs $r$ i.i.d random trials where in each trial one column of $A$ is selected by the following random process: we throw a biased die with $d$ faces with each face corresponding to a column of $A$, where for $i = 1, ..., d$ the $i$-th face occurs with probability $p_i$. We select the column of $A$ that corresponds to the face we threw in the current trial. Finally, note that the running time of Algorithm 1 is dominated by the time required to compute the top-$k$ right singular vectors of the matrix $A$, which is at most $O\left(\min\{nd^2, n^2 d\}\right)$.

**Input:** $n \times d$ matrix $A$ ($n$ points, $d$ features), number of clusters $k$, parameter $\epsilon \in (0, 1)$.

1. Compute the top-$k$ right singular vectors of $A$, denoted by $V_k \in R^{d \times k}$.

2. Compute the (normalized) leverage scores $p_i$, for $i = 1, \ldots, d$,

$$p_i = \left\| (V_k)_{(i)} \right\|_2^2 / k.$$

3. Fix a sampling parameter $r = \Theta(k \log(k/\epsilon)/\epsilon^2)$.

4. For $t = 1, \ldots, r$ i.i.d random trials:

   - keep the $i$-th feature with probability $p_i$ and multiply it by the factor $(rp_i)^{-1/2}$.

5. Return the $n \times r$ matrix $\tilde{A}$ containing the selected (rescaled) features.

**Output:** $n \times r$ matrix $\tilde{A}$, with $r = \Theta(k \log(k/\epsilon)/\epsilon^2)$.

Algorithm 1: A randomized feature selection algorithm for the $k$-means clustering problem.

In order to theoretically evaluate the accuracy of our feature selection algorithm, and provide some a priori guarantees regarding the quality of the clustering after feature selection is performed, we chose to report results on the optimal value of the $k$-means clustering objective (the $F_{opt}$ of Definition 1). This metric of accuracy has been extensively used in the Theoretical Computer Science community in order to analyze approximation algorithms for the $k$-means clustering problem. In particular, existing constant factor or relative error approximation algorithms for $k$-means (see, for example, [21, 1] and references therein) invariably approximate $F_{opt}$.

Obviously, Algorithm 1 does not return a partition of the rows of $A$. In a practical setting, it would be employed as a preprocessing step. Then, an approximation algorithm for the $k$-means clustering problem would be applied on $\tilde{A}$ in order to determine the partition of the rows of $A$. In order to formalize our discussion, we borrow a definition from the approximation algorithms literature.

**Definition 2** [K-MEANS APPROXIMATION ALGORITHM]
*An algorithm is a "$\gamma$-approximation" for the $k$-means clustering problem ($\gamma \geq 1$) if it takes inputs $A$ and $k$, and returns an indicator matrix $X_\gamma$ that satisfies with probability at least $1 - \delta_\gamma$,*

$$\left\| A - X_\gamma X_\gamma^T A \right\|_F^2 \leq \gamma \min_{X \in \mathcal{X}} \left\| A - XX^T A \right\|_F^2. \tag{3}$$

*In the above $\delta_\gamma \in [0, 1)$ is the failure probability of the algorithm.*

Clearly, when $\gamma = 1$, then $X_\gamma$ is the optimal partition, which is a well-known NP-hard objective. If we allow $\gamma > 1$, then many approximation algorithms exist in the literature. For example, the work of [21], achieves $\gamma = 1 + \epsilon$, for some $\epsilon \in (0, 1]$ in time linear on the size of the input. Similarly, the $k$-means++ method of [1] achieves $\gamma = O(\log(k))$ using the popular Lloyd's algorithm and a sophisticated randomized seeding. Theorem 1 (see Section 4 for its proof) is our main quality-of-approximation result for our feature selection algorithm.

**Theorem 1** *Let the $n \times d$ matrix $A$ and the positive integer $k$ be the inputs of the $k$-means clustering problem. Let $\epsilon \in (0, 1)$, and run Algorithm 1 with inputs $A$, $k$, and $\epsilon$ in order to construct the $n \times r$ matrix $\tilde{A}$ containing the selected features, where $r = \Theta(k \log(k/\epsilon)/\epsilon^2)$.*

*If we run any $\gamma$-approximation algorithm ($\gamma \geq 1$) for the $k$-means clustering problem, whose failure probability is $\delta_\gamma$, on inputs $\tilde{A}$ and $k$, the resulting cluster indicator matrix $X_{\tilde{\gamma}}$ satisfies with probability at least $0.5 - \delta_\gamma$,*

$$\left\| A - X_{\tilde{\gamma}} X_{\tilde{\gamma}}^T A \right\|_F^2 \leq (1 + (1 + \epsilon)\gamma) \min_{X \in \mathcal{X}} \left\| A - XX^T A \right\|_F^2. \tag{4}$$

The failure probability of the above theorem can be easily reduced using standard boosting methods.

# 3 Related work

Feature selection has received considerable attention in the machine learning and data mining communities. A large number of different techniques appeared in prior work, addressing the feature selection within the context of both clustering and classification. Surveys include [13], as well as [14], which reports the results of the NIPS 2003 challenge in feature selection. Popular feature selection techniques include the Laplacian scores [16], the Fisher scores [9], or the constraint scores [33]. In this section, we opt to discuss only a family of feature selection methods that are closely related to the leverage scores of our algorithm. To the best of our knowledge, all previous feature selection methods come with no theoretical guarantees of the form that we describe here.

Given as input an $n \times d$ object-feature matrix $A$ and a positive integer $k$, feature selection for Principal Components Analysis (PCA) corresponds to the task of identifying a subset of $k$ columns from $A$ that capture essentially the same information as do the top $k$ principal components of $A$. Jolliffe [18] surveys various methods for the above task. Four of them (called $B1$, $B2$, $B3$, and $B4$ in [18]) employ the Singular Value Decomposition of $A$ in order to identify columns that are somehow correlated with its top $k$ left singular vectors. In particular, $B3$ employs exactly the leverage scores in order to greedily select the $k$ columns corresponding to the highest scores; no theoretical results are reported. An experimental evaluation of the methods of [18] on real datasets appeared in [19]. Another approach employing the matrix of the top $k$ right singular vectors of $A$ and a Procrustes-type criterion appeared in [20]. From an applications perspective, [30] employed the methods of [18] and [20] for gene selection in microarray data analysis. From a complementary viewpoint, feature selection for clustering seeks to identify those features that have the most discriminative power among the set of all features. Continuing the aforementioned line of research, many recent papers present methods that somehow employ the SVD of the input matrix in order to select discriminative features; see, for example, [23, 5, 25, 26]. Finally, note that employing the leverage scores in a randomized manner similar to Algorithm 1 has already been proven to be accurate for least-squares regression [8] and PCA [7, 2].

## 3.1 Connections with the SVD

A well-known property connects the SVD of a matrix and $k$-means clustering. Recall Definition 1, and notice that $X_{opt}X_{opt}^T A$ is a matrix of rank at most $k$. From the SVD optimality [11], we immediately get that (see section 4.1 for useful notation)

$$\|A_{\rho-k}\|_F^2 = \|A - A_k\|_F^2 \leq \left\|A - X_{opt}X_{opt}^T A\right\|_F^2 = F_{opt}. \tag{5}$$

A more interesting connection between the SVD and $k$-means appeared in [6]. If the $n \times d$ matrix $A$ is projected on the subspace spanned by its top $k$ left singular vectors, then the resulting $n \times k$ matrix $\hat{A} = U_k\Sigma_k$ corresponds to a mapping of the original $d$-dimensional space to the optimal $k$-dimensional space. This process is equivalent to feature extraction: the top $k$ left singular vectors (the columns of $U_k$) correspond to the constructed features ($\Sigma_k$ is a simple rescaling operator). Prior to the work of [6], it was empirically known that running $k$-means clustering algorithms on the low-dimensional matrix $\hat{A}$ was a viable alternative to clustering the high-dimensional matrix $A$. The work of [6] formally argued that if we let the cluster indicator matrix $\hat{X}_{opt}$ denote the optimal $k$-means partition on $\hat{A}$, i.e.,

$$\hat{X}_{opt} = \arg \min_{X \in \mathcal{X}} \left\|\hat{A} - XX^T\hat{A}\right\|_F^2, \tag{6}$$

then using this partition on the rows of the original matrix $A$ is a 2-approximation to the optimal partition, a.k.a.,

$$\left\|A - \hat{X}_{opt}\hat{X}_{opt}^T A\right\|_F^2 \leq 2 \min_{X \in \mathcal{X}} \left\|A - XX^T A\right\|_F^2. \tag{7}$$

The above result is the starting point of our work here. Indeed, we seek to replace the $k$ artificial features that are extracted via the SVD with a small number (albeit slightly larger than $k$) of actual features. On the positive side, an obvious advantage of feature selection vs. feature extraction is the immediate interpretability of the former. On the negative side, our approximation accuracy is slightly worse ($2 + \epsilon$, see Theorem 1 with $\gamma = 1$) and we need slightly more than $k$ features.

# 4 The proof of Theorem 1

This section gives the proof of Theorem 1. We start by introducing useful notation; then, we present a preliminary lemma and the proof itself.

## 4.1 Notation

Given an $n \times d$ matrix $A$, let $U_k \in \mathbb{R}^{n \times k}$ (resp. $V_k \in \mathbb{R}^{d \times k}$) be the matrix of the top $k$ left (resp. right) singular vectors of $A$, and let $\Sigma_k \in \mathbb{R}^{k \times k}$ be a diagonal matrix containing the top $k$ singular values of $A$. If we let $\rho$ be the rank of $A$, then $A_{\rho-k}$ is equal to $A - A_k$, with $A_k = U_k \Sigma_k V_k^T$. $\|A\|_F$ and $\|A\|_2$ denote the Frobenius and the spectral norm of a matrix $A$, respectively. $A^+$ denotes the pseudo-inverse of $A$ and $\|A^+\|_2 = \sigma_{max}(A^+) = 1/\sigma_{min}(A)$, where $\sigma_{max}(X)$ and $\sigma_{min}(X)$ denote the largest and the smallest non-zero singular values of a matrix $X$, respectively. A useful property of matrix norms is that for any two matrices $X$ and $Y$, $\|XY\|_F \leq \|X\|_F \|Y\|_2$ and $\|XY\|_F \leq \|X\|_2 \|Y\|_F$; this is a stronger version of the standard submultiplicavity property for matrix norms. We call $P$ a projector matrix if it is square and $P^2 = P$. We use $\mathbf{E}[y]$ to take the expectation of a random variable $y$ and $\mathbf{Pr}[e]$ to take the probability of a random event $e$. Finally, we abbreviate "independent identically distributed" to "i.i.d" and "with probability" to "w.p".

## 4.2 Sampling and rescaling matrices

We introduce a simple matrix formalism in order to conveniently represent the sampling and rescaling processes of Algorithm 1. Let $S$ be a $d \times r$ sampling matrix that is constructed as follows: $S$ is initially empty. For all $t = 1, \ldots, r$, in turn, if the $i$-th feature of $A$ is selected by the random sampling process described in Algorithm 1, then $e_i$ (a column vector of all-zeros, except for its $i$-th entry which is set to one) is appended to $S$. Also, let $D$ be a $r \times r$ diagonal rescaling matrix constructed as follows: $D$ is initially an all-zeros matrix. For all $t = 1, \ldots, r$, in turn, if the $i$-th feature of $A$ is selected, then the next diagonal entry of $D$ is set to $1/\sqrt{rp_i}$. Thus, by using the notation of this paragraph, Algorithm 1 outputs the matrix $\tilde{A} = ASD \in \mathbb{R}^{n \times r}$.

## 4.3 A preliminary lemma and sufficient conditions

Lemma 1 presented below gives upper and lower bounds for the largest and the smallest singular values of the matrix $V_k^T SD$, respectively. This also implies that $V_k^T SD$ has full rank. Finally, it argues that the matrix $ASD$ can be used to provide a very accurate approximation to the matrix $A_k$.

Lemma 1 provides four sufficient conditions for designing provably accurate feature selection algorithms for $k$-means clustering. To see this notice that, in the proof of eqn. (4) given below, the results of Lemma 1 are sufficient to prove our main theorem; the rest of the arguments apply to all sampling and rescaling matrices $S$ and $D$. Any feature selection algorithm, i.e. any sampling matrix $S$ and rescaling matrix $D$, that satisfy bounds similar to those of Lemma 1, can be employed to design a provably accurate feature selection algorithm for $k$-means clustering. The quality of such an approximation will be proportional to the tightness of the bounds of the three terms of Lemma 1 ($\|V_k^T SD\|_2$, $\|(V_k^T SD)^+\|_2$, and $\|E\|_F$). Where no rescaling is allowed in the selected features, the bottleneck in the approximation accuracy of a feature selection algorithm would be to find a sampling matrix $S$ such that *only* $\|(V_k^T S)^+\|_2$ is bounded from above. To see this notice that, in Lemma 1, for any $S$, $\|V_k^T S\|_2 \leq 1$, and (after applying the submultiplicavity property of Section 4.1 in eqn. 13) $\|E\|_F \leq \|(V_k^T S)^+\|_2 \|A - A_k\|$. It is worth emphasizing that the same factor $\|(V_k^T S)^+\|_2$ appeared to be the bottleneck in the design of provably accurate column-based low-rank approximations (see, for example, Theorem 1.5 in [17] and eqn. (3.19) in [12]). It is evident from the above observations that other column sampling methods (see, for example, [17, 3, 2] and references therein), satisfying similar bounds to those of Lemma 1, immediately suggest themselves for the design of provably accurate feature selection algorithms for $k$-means clustering. Finally, equations (101) and (102) of Lemma 4.4 in [31] suggest that a sub-sampled randomized Fourier transform can be used for the design of a provably accurate feature extraction algorithm for k-means clustering, since they provide bounds similar to those of Lemma 1 by replacing the matrices $S$ and $D$ of our algorithm with a sub-sampled randomized Fourier transform matrix (see the matrix $\mathcal{R}$ of eqn. (6) in [31]).

**Lemma 1** *Assume that the sampling matrix $S$ and the rescaling matrix $D$ are constructed using Algorithm 1 (see also Section 4.2) with inputs $A$, $k$, and $\epsilon \in (0,1)$. Let $c_o$ and $c_1$ be absolute constants that will be specified later. If the sampling parameter $r$ of Algorithm 1 satisfies*

$$r \geq 2c_1 c_o^2 k \log(c_1 c_o^2 k/\epsilon^2)/\epsilon^2,$$

*then all four statements below hold together with probability at least $0.5$:*

1. *$\left\| V_k^T SD \right\|_2 = \sigma_{max}(V_k^T SD) \leq \sqrt{1+\lambda}$.*

2. *$\left\| (V_k^T SD)^+ \right\|_2 = 1/\sigma_{min}(V_k^T SD) \leq \sqrt{1/(1-\lambda)}$.*

3. *$V_k^T SD$ is a full rank matrix, i.e. $rank(V_k^T SD) = k$.*

4. *$A_k = (ASD)(V_k^T SD)^+ V_k^T + E$, with $\|E\|_F \leq \mu \|A - A_k\|_F$.*

*To simplify notation, we set $\lambda = \epsilon\sqrt{36/c_1}$ and $\mu = \epsilon\sqrt{6/(2c_1 c_o^2 \log(c_1 c_o^2 k/\epsilon^2))} + \sqrt{6\lambda^2/(1-\lambda)}$.*

*Proof:* First, we will apply Theorem 3.1 of [29] for an appropriate random vector $y$. Toward that end, for $i = 1, ..., d$, the $i$-th column of the matrix $V_k^T$ is denoted by $(V_k^T)^{(i)}$. We define the random vector $y \in R^k$ as follows: for $i = 1, ..., d$ $\mathbf{Pr}[y = y_i] = p_i$, where $y_i = (1/\sqrt{p_i})(V_k^T)^{(i)}$ is a realization of $y$. This definition of $y$ and the definition of the sampling and rescaling matrices $S$ and $D$ imply that $V_k^T SDDS^T V_k = \frac{1}{r}\sum_{i=1}^{d} y_i y_i^T$. Our choice of $p_i = \|(V_k^T)^{(i)}\|_2/k$ implies that $\|y\|_2 \leq \sqrt{k}$. Note also that $E[yy^T] = \sum_{i=1}^{d} p_i \frac{1}{\sqrt{p_i}}(V_k^T)^{(i)} \frac{1}{\sqrt{p_i}}(V_k^T)^{(i)T} = V_k^T V_k = I_k$. Obviously, $\|\mathbf{E}[yy^T]\|_2 = 1$. Our choice of $r$ allows us to apply Theorem 3.1 of [29], which, combined with the Markov's inequality on the random variable $z = \left\| V_k^T SDDS^T V_k - I_k \right\|_2$ implies that w.p at least $1 - 1/6$,

$$\left\| V_k^T SDDS^T V_k - I_k \right\|_2 \leq 6c_0\sqrt{k\log(r)/r},$$

for a sufficiently large (unspecified in [29]) constant $c_o$. Standard matrix perturbation theory results [11] imply that for $i = 1, ..., k$

$$\left\| V_k^T SDDS^T V_k - I_k \right\|_2 = \left| \sigma_i^2\left( V_k^T SD \right) - 1 \right| \leq 6c_o\sqrt{k\log(r)/r}.$$

Our choice of $r$ and simple algebra suffices to show that $\log(r)/r \leq \epsilon^2/(c_1 c_o^2 k)$, which implies that the first two statements of the Lemma hold w.p at least $1 - 5/6$. To prove the third statement, we only need to show that the $k$-th singular value of $V_k^T SD$ is positive. Our choice of $\epsilon \in (0,1)$ and the second condition of the Lemma imply that $\sigma_k(V_k^T SD) > 0$. To prove the fourth statement:

$$\left\| A_k - ASD(V_k^T SD)^+ V_k^T \right\|_F = \left\| A_k - A_k SD(V_k^T SD)^+ V_k^T - A_{\rho-k}SD(V_k^T SD)^+ V_k^T \right\|_F \quad (8)$$

$$\leq \underbrace{\left\| A_k - A_k SD(V_k^T SD)^+ V_k^T \right\|_F}_{\theta_1} + \underbrace{\left\| A_{\rho-k}SD(V_k^T SD)^+ V_k^T \right\|_F}_{\theta_2} \quad (9)$$

In the above, in eqn. (8) we replaced $A$ by $A_k + A_{\rho-k}$, and in eqn. (9) we used the triangle inequality. The first term of eqn. (9) is bounded by

$$\theta_1 = \left\| A_k - U_k \Sigma_k V_k^T SD(V_k^T SD)^+ V_k^T \right\|_F \quad (10)$$

$$= \left\| A_k - U_k \Sigma_k I_k V_k^T \right\|_F = 0. \quad (11)$$

In the above, in eqn. (10) we replaced $A_k$ by $U_k \Sigma_k V_k^T$, and in eqn. (11) we set $(V_k^T SD)(V_k^T SD)^+ = I_k$, since $V_k^T SD$ is a rank-$k$ matrix w.p $1 - 5/6$. The second term of eqn. (9) is bounded by

$$\theta_2 = \left\| U_{\rho-k}\Sigma_{\rho-k}V_{\rho-k}^T SD(V_k^T SD)^+ V_k^T \right\|_F \quad (12)$$

$$\leq \left\| \Sigma_{\rho-k}V_{\rho-k}^T SD(V_k^T SD)^+ \right\|_F. \quad (13)$$

In the above, in eqn. (12) we replaced $A_{\rho-k}$ by $U_{\rho-k}\Sigma_{\rho-k}V_{\rho-k}^T$, and in eqn. (13) $U_{\rho-k}$ and $V_k^T$ can be dropped without increasing a unitarily invariant norm such as the Frobenius matrix norm. If the first three statements of the lemma hold w.p at least $1 - 5/6$, then w.p at least $1 - 1/3$,

$$\left\| \Sigma_{\rho-k}V_{\rho-k}^T SD(V_k^T SD)^+ \right\|_F \leq \left( \epsilon\sqrt{6/(2c_1 c_o^2 \log(c_1 c_o^2 k/\epsilon^2))} + \sqrt{6\lambda^2/(1-\lambda)} \right) \|A - A_k\|_F.$$

(The proof of this last argument is omitted from this extended abstract.) Finally, notice that the first three statements have the same failure probability $1/6$ and the fourth statement fails w.p $1/3$; the union bound implies that all four statements hold together with probability at least $0.5$.

$\diamond$

## 4.4 The proof of eqn. (4) of Theorem 1

We assume that Algorithm 1 fixes $r$ to the value specified in Lemma 1; note that this does not violate the asymptotic notation used in Algorithm 1. We start by manipulating the term $\left\| A - X_{\tilde{\gamma}} X_{\tilde{\gamma}}^T A \right\|_F^2$ in eqn. (4). Replacing $A$ by $A_k + A_{\rho-k}$, and using the Pythagorean theorem (the subspaces spanned by the components $A_k - X_{\tilde{\gamma}} X_{\tilde{\gamma}}^T A_k$ and $A_{\rho-k} - X_{\tilde{\gamma}} X_{\tilde{\gamma}}^T A_{\rho-k}$ are perpendicular) we get

$$\left\| A - X_{\tilde{\gamma}} X_{\tilde{\gamma}}^T A \right\|_F^2 = \underbrace{\left\| (I - X_{\tilde{\gamma}} X_{\tilde{\gamma}}^T) A_k \right\|_F^2}_{\theta_3^2} + \underbrace{\left\| (I - X_{\tilde{\gamma}} X_{\tilde{\gamma}}^T) A_{\rho-k} \right\|_F^2}_{\theta_4^2}. \tag{14}$$

We first bound the second term of eqn. (14). Since $I - X_{\tilde{\gamma}} X_{\tilde{\gamma}}^T$ is a projector matrix, it can be dropped without increasing a unitarily invariant norm. Now eqn. (5) implies that

$$\theta_4^2 \leq F_{opt}. \tag{15}$$

We now bound the first term of eqn. (14):

$$\theta_3 \leq \left\| (I - X_{\tilde{\gamma}} X_{\tilde{\gamma}}^T) ASD(V_k SD)^+ V_k^T \right\|_F + \|E\|_F \tag{16}$$

$$\leq \left\| (I - X_{\tilde{\gamma}} X_{\tilde{\gamma}}^T) ASD \right\|_F \left\| (V_k SD)^+ \right\|_2 + \|E\|_F \tag{17}$$

$$\leq \sqrt{\gamma} \left\| (I - X_{opt} X_{opt}^T) ASD \right\|_F \left\| (V_k SD)^+ \right\|_2 + \|E\|_F \tag{18}$$

$$\leq \sqrt{\gamma} \left\| (I - X_{opt} X_{opt}^T) ASD(V_k SD)^+ \right\|_F \left\| (V_k SD) \right\|_2 \left\| (V_k SD)^+ \right\|_2 + \|E\|_F \tag{19}$$

$$= \sqrt{\gamma} \underbrace{\left\| (I - X_{opt} X_{opt}^T) ASD(V_k SD)^+ V_k^T \right\|_F}_{\theta_5} \left\| (V_k SD) \right\|_2 \left\| (V_k SD)^+ \right\|_2 + \|E\|_F \tag{20}$$

In eqn. (16) we used Lemma 1, the triangle inequality, and the fact that $I - \tilde{X}_{\gamma} \tilde{X}_{\gamma}^T$ is a projector matrix and can be dropped without increasing a unitarily invariant norm. In eqn. (17) we used submultiplicativity (see Section 4.1) and the fact that $V_k^T$ can be dropped without changing the spectral norm. In eqn. (18) we replaced $X_{\tilde{\gamma}}$ by $X_{opt}$ and the factor $\sqrt{\gamma}$ appeared in the first term. To better understand this step, notice that $X_{\tilde{\gamma}}$ gives a $\gamma$-approximation to the optimal $k$-means clustering of the matrix $ASD$, and any other $n \times k$ indicator matrix (for example, the matrix $X_{opt}$) satisfies

$$\left\| (I - X_{\tilde{\gamma}} X_{\tilde{\gamma}}^T) ASD \right\|_F^2 \leq \gamma \min_{X \in \mathcal{X}} \left\| (I - XX^T) ASD \right\|_F^2 \leq \gamma \left\| (I - X_{opt} X_{opt}^T) ASD \right\|_F^2.$$

In eqn. (19) we first introduced the $k \times k$ identity matrix $I_k = (V_k^T SD)^+ (V_k^T SD)$ $(rank(V_k^T SD) = k)$ and then we used submultiplicativity (see Section 4.1). In eqn. (20) we introduced $V_k^T$ without changing the Frobenius norm. We further manipulate the term $\theta_5$ of eqn. (20):

$$\theta_5 \leq \left\| (I - X_{opt} X_{opt}^T) A_k \right\|_F + \left\| (I - X_{opt} X_{opt}^T) E \right\|_F \tag{21}$$

$$\leq \left\| (I - X_{opt} X_{opt}^T) AV_k V_k^T \right\|_F + \|E\|_F \tag{22}$$

$$\leq (1 + \mu) \sqrt{F_{opt}} \tag{23}$$

In eqn. (21) we used Lemma 1 and the triangle inequality. In eqn. (22) we replaced $A_k$ by $AV_k V_k^T$ and dropped $I - X_{opt} X_{opt}^T$ from the second term ($I - X_{opt} X_{opt}^T$ is a projector matrix and does not increase the Frobenius norm). In eqn. (23) we dropped the projector matrix $V_k V_k^T$ and used eqn. (5) and Definition 1. Combining equations (20), (23), (5), Lemma 1, and the fact that $\gamma \geq 1$, we get

$$\theta_3 \leq \sqrt{\gamma} \underbrace{(\sqrt{\frac{1 + \lambda}{1 - \lambda}} (1 + \mu) + \mu)}_{\theta_6} \sqrt{F_{opt}}.$$

Simple algebra suffices to show that for any $\epsilon \in (0, 1)$, for any positive integer $k \geq 1$, and for some sufficiently large constant $c_1$, it is

$$\theta_6 \leq \sqrt{1 + \epsilon},$$

thus

$$\theta_3^2 \leq \gamma(1 + \epsilon) F_{opt}. \tag{24}$$

Combining eqn. (24) with eqns. (14) and (15) concludes the proof of eqn. (4). Using asymptotic notation our choice of $r$ satisfies $r = \Omega(k \log(k/\epsilon)/\epsilon^2)$. Note that Theorem 1 fails only if Lemma 1 or the $\gamma$-approximation $k$-means clustering algorithm fail, which happens w.p at most $0.5 + \delta_{\gamma}$.

| | r = 5k | | r = 10k | | r = 20k | | All | |
|---|---|---|---|---|---|---|---|---|
| | $P$ | $F$ | $P$ | $F$ | $P$ | $F$ | $P$ | $F$ |
| **NIPS** ($k = 3$) | .847 | .758 | .847 | .751 | .859 | .749 | .881 | .747 |
| **Bio** ($k = 3$) | .742 | .764 | .935 | 0.726 | 1 | .709 | 1 | .709 |

Table 1: Numerics from our experiments (Leverage scores).

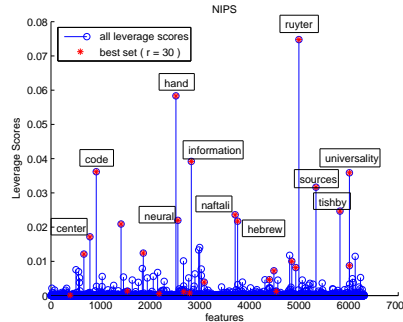

Figure 1: Leverage scores for the NIPS dataset.

## 5 Empirical study

We present an empirical evaluation of Algorithm 1 on two real datasets. We show that it selects the most relevant features (Figure 1) and that the clustering obtained after feature selection is performed is very accurate (Table 1). It is important to note that the choice of $r$ in the description of Algorithm 1 is a sufficient - not necessary - condition to prove our theoretical bounds. Indeed, a much smaller choice of $r$, for example $r = 10k$, is often sufficient for good empirical results.

We first experimented with a NIPS documents dataset (see `http://robotics.stanford.edu/~gal/` and [10]). The data consist of a $184 \times 6314$ document-term matrix $A$, with $A_{ij}$ denoting the number of occurrences of the $j$-th term in the $i$-th document. Each document is a paper that appeared in the proceedings of NIPS 2001, 2002, or 2003, and belongs to one of the following three topic categories: (i) Neuroscience, (ii) Learning Theory, and (iii) Control and Reinforcement Learning. Each term appeared at least once in one of the $184$ documents. We evaluated the accuracy of Algorithm 1 by running the Lloyd's heuristic[1] on the rescaled features returned by our method. In order to drive down the failure probability of Algorithm 1, we repeated it 30 times (followed by the Lloyd' heuristic each time) and kept the partition that minimized the objective value. We report the percentage of correctly classified objects (denoted by $P$, $0 \leq P \leq 1$), as well as the value of the $k$-means objective (i.e., the value $F = ||A - X_{\tilde{\gamma}} X_{\tilde{\gamma}}^T A||_F^2 / ||A||_F^2$ of Theorem 1; the division by the $||A||_F^2$ is for normalization). Results are depicted in Table 1. Notice that only a small subset of features suffices to approximately reproduce the partition obtained when all features were kept. In Figure 1 we plotted the distribution of the leverage scores for the 6314 terms (columns) of $A$; we also highlighted the features returned by Algorithm 1 when the sampling parameter $r$ is set to $10k$. We observed that terms corresponding to the largest leverage scores had significant discriminative power. In particular, `ruyter` appeared almost exclusively in documents of the first and third categories, `hand` appeared in documents of the third category, `information` appeared in documents of the first category, and `code` appeared in documents of the second and third categories only. We also experimented with microarray data showing the expression levels of 5520 genes (features) for 31 patients (objects) having three different cancer types [27]: 10 patients with gastrointestinal stromal tumor, 12 with leiomyosarcoma, and 9 with synovial sarcoma. Table 1 depicts the results from our experiments by choosing $k = 3$. Note that the Lloyd's heuristic worked almost perfectly when $r$ was set to $10k$ and perfectly when $r$ was set to $20k$. Experimental parameters set to the same values as in the first experiment.

## Footnotes

[1]We ran 30 iterations of the E-M step with 30 different random initializations and returned the partition that minimized the k-means objective function, i.e. we ran kmeans(A, k, 'Replicates', 30, 'Maxiter', 30) in MatLab.

# References

[1] D. Arthur and S. Vassilvitskii. k-means++: the advantages of careful seeding. In *Proceedings of the 18th Annual ACM-SIAM Symposium on Discrete algorithms (SODA)*, pages 1027–1035, 2007.

[2] C. Boutsidis, M. W. Mahoney, and P. Drineas. Unsupervised feature selection for Principal Components Analysis. In *Proceedings of the 14th Annual ACM SIGKDD Conference (KDD)*, pages 61–69, 2008.

[3] S. Chandrasekaran and I. Ipsen. On rank-revealing factorizations. *SIAM Journal on Matrix Analysis and Applications*, 15:592–622, 1994.

[4] S. Chatterjee and A. S. Hadi. Influential observations, high leverage points, and outliers in linear regression. *Statistical Science*, 1:379–393, 1986.

[5] Y. Cui and J. G. Dy. Orthogonal principal feature selection. *manuscript*.

[6] P. Drineas, A. Frieze, R. Kannan, S. Vempala, and V. Vinay. Clustering in large graphs and matrices. In *Proceedings of the 10th Annual ACM-SIAM Symposium on Discrete Algorithms (SODA)*, pages 291–299, 1999.

[7] P. Drineas, M. Mahoney, and S. Muthukrishnan. Relative-Error CUR Matrix Decompositions. *SIAM Journal on Matrix Analysis and Applications*, 30:844–881, 2008.

[8] P. Drineas, M. Mahoney, and S. Muthukrishnan. Sampling algorithms for $\ell_2$ regression and applications. In *Proceedings of the 17th Annual ACM-SIAM Symposium on Discrete Algorithms (SODA)*, pages 1127–1136, 2006.

[9] D. Foley and J. Sammon, J.W. An optimal set of discriminant vectors. *IEEE Transactions on Computers*, C-24(3):281–289, March 1975.

[10] A. Globerson, G. Chechik, F. Pereira, and N. Tishby. Euclidean Embedding of Co-occurrence Data. *The Journal of Machine Learning Research*, 8:2265–2295, 2007.

[11] G. Golub and C. V. Loan. *Matrix Computations*. Johns Hopkins University Press, Baltimore, 1989.

[12] S. A. Goreinov, E. E. Tyrtyshnikov, and N. L. Zamarashkin A theory of pseudoskeleton approximations. *Linear Algebra and Its Applications*, 261:1-21, 1997.

[13] I. Guyon and A. Elisseeff. An introduction to variable and feature selection. *Journal of Machine Learning Research*, 3:1157–1182, 2003.

[14] I. Guyon, S. Gunn, A. Ben-Hur, and G. Dror. Result analysis of the NIPS 2003 feature selection challenge. In *Advances in Neural Information Processing Systems (NIPS) 17*, pages 545–552, 2005.

[15] J.A. Hartigan. Clustering algorithms. John Wiley & Sons, Inc. New York, NY, USA, 1975.

[16] X. He, D. Cai, and P. Niyogi. Laplacian score for feature selection. In *Advances in Neural Information Processing Systems (NIPS) 18*, pages 507–514. 2006.

[17] Y.P. Hong and C.T. Pan. Rank-revealing QR factorizations and the singular value decomposition. Mathematics of Computation, 58:213232, 1992.

[18] I. Jolliffe. Discarding variables in a principal component analysis. I: Artificial data. *Applied Statistics*, 21(2):160–173, 1972.

[19] I. Jolliffe. Discarding variables in a principal component analysis. II: Real data. *Applied Statistics*, 22(1):21–31, 1973.

[20] W. Krzanowski. Selection of variables to preserve multivariate data structure, using principal components. *Applied Statistics*, 36(1):22–33, 1987.

[21] A. Kumar, Y. Sabharwal, and S. Sen. A simple linear time $(1 + \epsilon)$-approximation algorithm for k-means clustering in any dimensions. In *Proceedings of the 45th Annual IEEE Symposium on Foundations of Computer Science (FOCS)*, pages 454–462, 2004.

[22] S.P. Lloyd. Least squares quantization in PCM. Unpublished Bell Lab. Tech. Note, portions presented at the Institute of Mathematical Statistics Meeting Atlantic City, NJ, September 1957. Also, IEEE Trans Inform Theory (Special Issue on Quantization), vol IT-28, pages 129-137, March 1982.

[23] Y. Lu, I. Cohen, X. S. Zhou, and Q. Tian. Feature selection using principal feature analysis. In *Proceedings of the 15th international conference on Multimedia*, pages 301–304, 2007.

[24] M. W. Mahoney and P. Drineas. CUR Matrix Decompositions for Improved Data Analysis. In *Proceedings of the National Academy of Sciences, USA (PNAS)*, 106, pages 697-702, 2009.

[25] A. Malhi and R. Gao. PCA-based feature selection scheme for machine defect classification. *IEEE Transactions on Instrumentation and Measurement*, 53(6):1517–1525, Dec. 2004.

[26] K. Mao. Identifying critical variables of principal components for unsupervised feature selection. *IEEE Transactions on Systems, Man, and Cybernetics*, 35(2):339–344, April 2005.

[27] T. Nielsen et al. Molecular characterisation of soft tissue tumors: A gene expression study. *Lancet*, 359:1301–1307, 2002.

[28] R. Ostrovsky, Y. Rabani, L. J. Schulman, and C. Swamy. The effectiveness of Lloyd-type methods for the k-means problem. In *Proceedings of the 47th Annual IEEE Symposium on Foundations of Computer Science (FOCS)*, pages 165–176, 2006.

[29] M. Rudelson, and R. Vershynin, Sampling from large matrices: An approach through geometric functional analysis. *Journal of the ACM (JACM)*, 54(4), July 2007.

[30] A. Wang and E. A. Gehan. Gene selection for microarray data analysis using principal component analysis. *Stat Med*, 24(13):2069–2087, July 2005.

[31] F. Woolfe, E. Liberty, V. Rokhlin, and M. Tygert. A fast randomized algorithm for the approximation of matrices. *Applied and Computational Harmonic Analysis*, 25 (3): 335-366, 2008.

[32] X. Wu et al. Top 10 algorithms in data mining analysis. *Knowl. Inf. Syst.*, 14(1):1–37, 2007.

[33] D. Zhang, S. Chen, and Z.-H. Zhou. Constraint score: A new filter method for feature selection with pairwise constraints. *Pattern Recognition*, 41(5):1440–1451, 2008.

